# Computing regularization paths for learning multiple kernels

**Francis R. Bach & Romain Thibaux**
Computer Science
University of California
Berkeley, CA 94720
*{fbach,thibaux}@cs.berkeley.edu*

**Michael I. Jordan**
Computer Science and Statistics
University of California
Berkeley, CA 94720
*jordan@cs.berkeley.edu*

## Abstract

The problem of learning a sparse conic combination of kernel functions or kernel matrices for classification or regression can be achieved via the regularization by a block 1-norm [1]. In this paper, we present an algorithm that computes the entire regularization path for these problems. The path is obtained by using numerical continuation techniques, and involves a running time complexity that is a constant times the complexity of solving the problem for one value of the regularization parameter. Working in the setting of kernel linear regression and kernel logistic regression, we show empirically that the effect of the block 1-norm regularization differs notably from the (non-block) 1-norm regularization commonly used for variable selection, and that the regularization path is of particular value in the block case.

## 1 Introduction

Kernel methods provide efficient tools for nonlinear learning problems such as classification or regression. Given a learning problem, two major tasks faced by practitioners are to find an appropriate kernel and to understand how regularization affects the solution and its performance. This paper addresses both of these issues within the supervised learning setting by combining three themes from recent statistical machine learning research, namely multiple kernel learning [2, 3, 1], computation of regularization paths [4, 5], and the use of path following methods [6].

The problem of learning the kernel from data has recently received substantial attention, and several formulations have been proposed that involve optimization over the conic structure of the space of kernels [2, 1, 3]. In this paper we follow the specific formulation of [1], who showed that learning a conic combination of basis kernels is equivalent to regularizing the original supervised learning problem by a weighted block 1-norm (see Section 2.2 for further details). Thus, by solving a single convex optimization problem, the coefficients of the conic combination of kernels and the values of the parameters (the dual variables) are obtained. Given the basis kernels and their coefficients, there is one free parameter remaining—the regularization parameter.

Kernel methods are nonparametric methods, and thus regularization plays a crucial role in their behavior. In order to understand a nonparametric method, in particular complex non-

parametric methods such as those considered in this paper, it is useful to be able to consider the entire path of regularization, that is, the set of solutions for all values of the regularization parameter [7, 4]. Moreover, if it is relatively cheap computationally to compute this path, then it may be of practical value to compute the path as standard practice in fitting a model. This would seem particularly advisable in cases in which performance can display local minima along the regularization path. In such cases, standard local search methods may yield unnecessarily poor performance.

For least-squares regression with a 1-norm penalty or for the support vector machine, there exist efficient computational techniques to explore the regularization path [4, 5]. These techniques exploit the fact that for these problems the path is piecewise linear. In this paper we consider the extension of these techniques to the multiple kernel learning problem. As we will show (in Section 3), in this setting the path is no longer piecewise linear. It is, however, piecewise smooth, and we are able to follow it by using numerical continuation techniques [8, 6]. To do this in a computationally efficient way, we invoke logarithmic barrier techniques analogous to those used in interior point methods for convex optimization (see Section 3.3). As we shall see, the complexity of our algorithms essentially depends on the number of "kinks" in the path, i.e., the number of discontinuity points of the derivative. Our experiments suggest that the number of those kinks is always less than a small constant times the number of basis kernels. The empirical complexity of our algorithm is thus a constant times the complexity of solving the problem using interior point methods for one value of the regularization parameter (see Section 3.4 for details).

In Section 4, we present simulation experiments for classification and regression problems, using a large set of basis kernels based on the most widely used kernels (linear, polynomial, Gaussian). In particular, we show empirically that the number of kernels in the conic combination is not a monotonic function of the amount of regularization. This contrasts with the simpler non-block 1-norm case for variable selection (i.e., blocks of size one [4]), where the number of variables is usually monotonic (or nearly so). Thus the need to compute full regularization paths is particularly acute in our more complex (block 1-norm regularization) case.

## 2 Block 1-norm regularization

In this section we review the block 1-norm regularization framework of [1] as it applies to differentiable loss functions. To provide necessary background we begin with a short review of classical 2-norm regularization.

### 2.1 Classical 2-norm regularization

**Primal formulation**  We consider the general regularized learning optimization problem [7], where the data $x_i$, $i = 1, \ldots, n$, belong to the *input space* $\mathcal{X}$, and $y_i$, $i = 1, \ldots, n$ are the *responses* (lying either in $\{-1, 1\}$ for classification or $\mathbb{R}$ for regression). We map the data into a *feature space* $\mathcal{F}$ through $x \mapsto \Phi(x)$. The kernel associated with this feature map is denoted $k(x, y) = \Phi(x)^\top \Phi(y)$. The optimization problem is the following[1]:

$$\min_{w \in \mathbb{R}^p} \sum_{i=1}^n \ell(y_i, w^\top \Phi(x_i)) + \frac{\lambda}{2} ||w||^2, \tag{1}$$

where $\lambda > 0$ is a regularization parameter and $||w||$ is the 2-norm of $w$, defined as $||w|| = (w^\top w)^{1/2}$. The loss function $\ell$ is any function from $\mathbb{R} \times \mathbb{R}$ to $\mathbb{R}$. In this paper, we focus on loss functions that are strictly convex and twice continuously differentiable in their second argument. Let $\psi_i(v)$, $v \in \mathbb{R}$, be the Fenchel conjugate [9] of the convex function $\varphi_i(u) = \ell(y_i, u)$, defined as $\psi_i(v) = \max_{u \in \mathbb{R}}(vu - \varphi_i(u))$. Since we have assumed that

$\ell$ is strictly convex and differentiable, the maximum defining $\psi_i(v)$ is attained at a unique point equal to $\psi_i'(v)$ (possibly equal to $+\infty$ or $-\infty$). The function $\psi_i(v)$ is then strictly convex and twice differentiable in its domain.

In particular, we have the following examples in mind: for *least-squares regression*, we have $\varphi_i(u) = \frac{1}{2}(y_i - u)^2$ and $\psi_i(v) = \frac{1}{2}v^2 + vy_i$, while for *logistic regression*, we have $\varphi_i(u) = \log(1 + \exp(-y_i u_i))$, where $y_i \in \{-1, 1\}$, and $\psi_i(v) = (1 + vy_i)\log(1 + vy_i) - vy_i \log(-vy_i)$ if $vy_i \in (-1, 0)$, $+\infty$ otherwise.

**Dual formulation and optimality conditions** The Lagrangian for problem (1) is

$$\mathcal{L}(w, u, \alpha) = \sum_i \varphi_i(u_i) + \frac{\lambda}{2}||w||^2 - \lambda \sum_i \alpha_i(u_i - w^\top \Phi(x_i))$$

and is minimized with respect to $u$ and $w$ with $w = -\sum_i \alpha_i \Phi(x_i)$. The dual problem is then

$$\max_{\alpha \in \mathbb{R}^n} \left( -\sum_i \psi_i(\lambda \alpha_i) - \frac{\lambda}{2}\alpha^\top K \alpha \right), \tag{2}$$

where $K \in \mathbb{R}^{n \times n}$ is the kernel matrix of the points, i.e., $K_{ab} = k(x_a, x_b)$. The optimality condition for the dual variable $\alpha$ is then:

$$\forall i, (K\alpha)_i + \psi_i'(\lambda \alpha_i) = 0 \tag{3}$$

## 2.2 Block 1-norm regularization

In this paper, we map the input space $\mathcal{X}$ to $m$ different feature spaces $\mathcal{F}_1, \ldots, \mathcal{F}_m$, through $m$ feature maps $\Phi_1(x), \ldots, \Phi_m(x)$. We now have $m$ different variables $w_j \in \mathcal{F}_j$, $j = 1, \ldots, m$. We use the notation $\Phi(x) = (\Phi_1(x), \ldots, \Phi_m(x))$ and $w = (w_1, \ldots, w_m)$, and from now on, we use the implicit convention that the index $i$ ranges over data points (from 1 to $n$), while the index $j$ ranges over kernels/feature spaces (from 1 to $m$).

Let $d_j$, $j = 1, \ldots, m$, be weights associated with each kernel. We will see in Section 4 how these should be linked to the rank of the kernel matrices. Following [1], we consider the following problem with weighted block 1-norm regularization[2] (where $||w_j|| = (w_j^\top w_j)^{1/2}$ still denotes the 2-norm of $w_j$):

$$\min_{w \in \mathcal{F}_1 \times \cdots \times \mathcal{F}_m} \sum_i \varphi_i(w^\top \Phi(x_i)) + \lambda \sum_j d_j ||w_j||. \tag{4}$$

The problem (4) is a convex problem, but not differentiable. In order to derive optimality conditions, we can reformulate it with conic constraints and derive the following dual problem (we omit details for brevity) [9, 1]:

$$\max_\alpha - \sum_i \psi_i(\lambda \alpha_i) \text{ such that } \forall j, \alpha^\top K_j \alpha \leqslant d_j^2 \tag{5}$$

where $K_j$ is the *kernel matrix* associated with kernel $k_j$, i.e., defined as $(K_j)_{ab} = k_j(x_a, x_b)$. From the KKT conditions for problem Eq. (5), we obtain that the dual variable $\alpha$ is optimal if and only if there exists $\eta \in \mathbb{R}^m$ such that $\eta \geqslant 0$ and

$$\forall i, (\sum_j \eta_j K_j \alpha)_i + \psi_i'(\lambda \alpha_i) = 0 \tag{6}$$

$$\forall j, \alpha^\top K_j \alpha \leqslant d_j^2, \eta_j \geqslant 0, \eta_j(d_i^2 - \alpha^\top K_j \alpha) = 0.$$

We can go back and forth between optimal $w$ and $\alpha$ by $w = -\lambda \operatorname{Diag}(\eta) \sum_i \alpha_i x_i$ or $\alpha_i = \frac{1}{\lambda}\varphi_i'(w^\top x_i)$.

We see that the solution of Eq. (5) can be obtained by using only the kernel matrices $K_j$ (i.e., this is indeed a kernel machine) and that the optimal solution of the block 1-norm

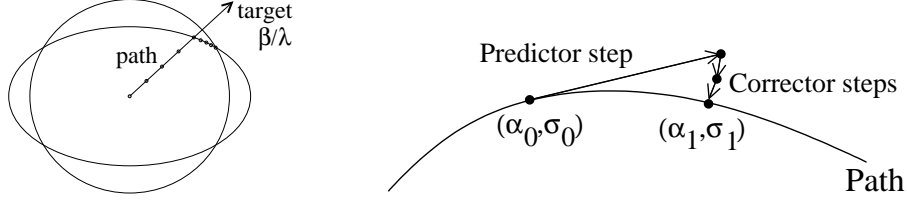

Figure 1: (Left) Geometric interpretation of the dual problem in Eq. (5) for linear regression; see text for details. (Right) Predictor-corrector algorithm.

problem in Eq. (5), with optimality conditions in Eq. (6), is the solution of the regular 2-norm problem in Eq. (2) with kernel $K = \sum_j \eta_j K_j$. Thus, with this formulation, we learn the coefficients of the conic combination of kernels as well as the dual variables $\alpha$ [1]. As shown in [1], the conic combination is sparse, i.e., many of the coefficients $\eta_j$ are equal to zero.

### 2.3 Geometric interpretation of dual problem

Each function $\psi_i$ is strictly convex, with a strict minimum at $\beta_i$ defined by $\psi_i'(\beta_i) = 0$ (for least-squares regression we have $\beta_i = -y_i$, and for the logistic regression we have $\beta_i = -y_i/2$). The negated dual objective $\sum_i \psi_i(\lambda \alpha_i)$ is thus a metric between $\alpha$ and $\beta/\lambda$ (for least-squares regression, this is simply the squared distance while for logistic regression, this is an entropy distance). Therefore, the dual problem aims to minimize a metric between $\alpha$ and the *target* $\beta/\lambda$, under the constraint that $\alpha$ belongs to an intersection of $m$ ellipsoids $\{\alpha \in \mathbb{R}^n, \alpha^\top K_j \alpha \leqslant d_j^2\}$.

When computing the regularization path from $\lambda = +\infty$ to $\lambda = 0$, the target goes from 0 to $\infty$ in the direction $\beta$ (see Figure 1). The geometric interpretation immediately implies that as long as $\frac{1}{\lambda^2}\beta^\top K_j \beta \leqslant d_j^2$, the active set is empty, the optimal $\alpha$ is equal to $\beta/\lambda$ and the optimal $w$ is equal to 0. We thus initialize the path following technique with $\lambda = \max_j(\beta^\top K_j \beta/d_j^2)^{1/2}$ and $\alpha = \beta/\lambda$.

## 3 Building the regularization path

In this section, the goal is to vary $\lambda$ from $+\infty$ (no regularization) to 0 (full regularization) and obtain a representation of the path of solutions $(\alpha(\lambda), \eta(\lambda))$. We will essentially approximate the path by a piecewise linear function of $\sigma = \log(\lambda)$.

### 3.1 Active set method

For the dual formulation Eq. (5)-Eq. (6), if the set of active kernels $\mathcal{J}(\alpha)$ is known, i.e., the set of kernels that are such that $\alpha^\top K_j \alpha = d_j^2$, then the optimality conditions become

$$\forall j \in \mathcal{J}, \alpha^\top K_j \alpha = d_j^2 \tag{7}$$
$$\forall i, \ \left(\sum_{j \in \mathcal{J}} \eta_j K_j \alpha\right)_i + \psi_i'(\lambda \alpha_i) = 0$$

and they are valid as long as $\forall j \notin \mathcal{J}, \alpha^\top K_j \alpha \leqslant d_j^2$ and $\forall j \in \mathcal{J}, \eta_j \geqslant 0$.

The path is thus piecewise smooth, with "kinks" at each point where the active set $\mathcal{J}$ changes. On each of the smooth sections, only those kernels with index belonging to $\mathcal{J}$ are used to define $\alpha$ and $\eta$, through Eq. (7). When all blocks have size one, or equivalently when all kernel matrices have rank one, then the path is provably linear in $1/\lambda$ between each kink [4] and is thus easy to follow. However, when the kernel matrices have higher

rank, this is not the case and additional numerical techniques are needed, which we now present. In the regularized formulation we present in Section 3.3, the optimal $\eta$ is a function of $\alpha$, and therefore we only have to follow the optimal $\alpha$, as a function of $\sigma = \log(\lambda)$.

## 3.2 Following a smooth path using numerical continuation techniques

In this section, we provide a brief review of path following, focusing on predictor-corrector methods [8]. We assume that the function $\alpha(\sigma) \in \mathbb{R}^d$ is defined implicitly by $J(\alpha, \sigma) = 0$, where $J$ is $C^\infty$ from $\mathbb{R}^{d+1}$ to $\mathbb{R}^d$ and $\sigma$ is a real variable. Starting from a point $\alpha_0, \sigma_0$ such that $J(\alpha_0, \sigma_0) = 0$, by the implicit function theorem, the solution is well defined and $C^\infty$ if the differential $\frac{\partial J}{\partial \alpha} \in \mathbb{R}^{d \times d}$ is invertible. The derivative at $\sigma_0$ is then equal to $\frac{d\alpha}{d\sigma}(\sigma_0) = -\left(\frac{\partial J}{\partial \alpha}(\alpha_0, \sigma_0)\right)^{-1} \frac{\partial J}{\partial \sigma}(\alpha_0, \sigma_0)$.

In order to follow the curve $\alpha(\sigma)$, the most effective numerical method is the predictor-corrector method, which works as follows (see Figure 1):

- *predictor step* : from $(\alpha_0, \sigma_0)$ predict where $\alpha(\sigma_0 + h)$ should be using the first order expansion, i.e., take $\lambda_1 = \lambda_0 + h$, $\alpha_1 = \alpha_0 + h\frac{d\alpha}{d\sigma}(\sigma_0)$ (note that $h$ can be chosen positive or negative, depending on the direction we want to follow).

- *corrector steps* : $(\alpha_1, \sigma_1)$ might not satisfy $J(\alpha_1, \sigma_1) = 0$, i.e., the tangent prediction might (and generally will) leave the curve $\alpha(\sigma)$. In order to return to the curve, Newton's method is used to solve the nonlinear system of equations (in $\alpha$) $J(\alpha, \sigma_1) = 0$, starting from $\alpha = \alpha_1$. If $h$ is small enough, then the Newton steps will converge quadratically to a solution $\alpha_2$ of $J(\alpha, \sigma_1) = 0$ [8].

Methods that do only one of the two steps are not as efficient: doing only predictor steps is not stable and the algorithm leaves the path very quickly, whereas doing only corrector steps (with increasing $\sigma$) is essentially equivalent to seeding the optimizer for a given $\sigma$ with the solution for a previous $\sigma$, which is very inefficient in sections where the path is close to linear. Predictor-corrector methods approximate the path by a sequence of points on that path, which can be joined to provide a piecewise linear approximation.

At first glance, in order to follow the piecewise smooth path all that is needed is to follow each piece and detect when the active set changes, i.e, when $\exists j \notin \mathcal{J}, \alpha^\top K_j \alpha = d_j^2$ or $\exists j \in \mathcal{J}, \eta_j = 0$. However this approach can be tricky numerically [8]. We instead prefer to use a numerical regularization technique that will (a) make the entire path smooth, (b) make sure that the Newton steps are globally convergent, and (c) will still enable us to use only a subset of the kernels to define the path locally.

## 3.3 Numerical regularization

We borrow a classical regularization method from interior point methods, in which a constrained problem is made unconstrained by using a convex log-barrier [9]. In the dual formulation, we solve the following problem (note that we now use a min-problem and we have divided by $\lambda^2$, which leaves the problem unchanged), where $\mu$ is a fixed small constant:

$$\min_\alpha F(\alpha, \lambda) \text{ where } F(\alpha, \lambda) = \sum_i \frac{1}{\lambda^2}\psi_i(\lambda\alpha_i) - \frac{\mu}{2\lambda}\sum_j \log(d_j^2 - \alpha^\top K_j \alpha) \quad (8)$$

For $\lambda$ fixed, $\alpha \mapsto F(\alpha, \lambda)$ is $C^\infty$ and strictly convex in its domain $\{\alpha, \forall j, \alpha^\top K_j \alpha < d_j^2\}$, and thus the global minimum is uniquely defined by $\frac{\partial F}{\partial \alpha} = 0$. If we define $\eta_j(\alpha) = \mu/(d_j^2 - \alpha^\top K_j \alpha)$, then we have $\frac{\partial F}{\partial \alpha_i} = \frac{1}{\lambda}\psi_i'(\lambda\alpha_i) + \frac{1}{\lambda}\sum_j \eta_j(\alpha)(K_j\alpha)_i$, and thus, the optimality condition for the problem with the log-barrier is exactly equivalent to the one in Eq. (6). But now instead of having $\eta_j(d_j^2 - \alpha^\top K_j \alpha) = 0$ (which would define an optimal solution of the numerically unregularized problem), we have $\eta_j(d_j^2 - \alpha^\top K_j \alpha) = \mu$. Any

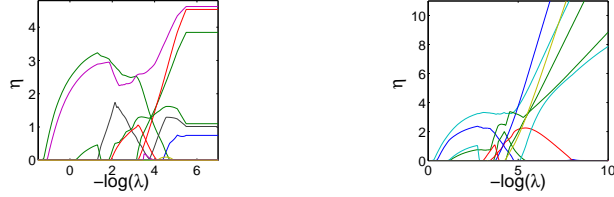

Figure 2: Examples of variation of $\eta$ along the regularization path for linear regression (left) and logistic regression (right).

dual-feasible variables $\eta$ and $\alpha$ (not necessarily linked through a functional relationship) define primal-dual variables and the quantity $\eta_j (d_j^2 - \alpha^\top K_j \alpha)$ is exactly the *duality gap* [9], i.e., the difference between the primal and dual objectives. Thus the parameter $\mu$ holds fixed the duality gap we are willing to pay. In simulations, we used $\mu = 10^{-3}$.

We can apply the techniques of Section 3.2 to follow the path for a fixed $\mu$, for the variables $\alpha$ only, since $\eta$ is now a function of $\alpha$. The corrector steps, are not only Newton steps for solving a system of nonlinear equations, they are also Newton-Raphson steps to minimize a strictly convex function, and are thus globally convergent [9].

### 3.4 Path following algorithm

Our path following algorithm is simply a succession of predictor-corrector steps, described in Section 3.2, with $J(\alpha, \sigma) = \frac{\partial F}{\partial \alpha}(\alpha, \sigma)$ defined in Section 3.3, where $\sigma = \log(\lambda)$. The initialization presented in Section 2.3 is used.

In Figure 2, we show simple examples of the values of the kernel weights $\eta$ along the path for a toy problem with a small number of kernels, for kernel linear regression and kernel logistic regression. It is worth noting that the weights are not even approximately monotonic functions of $\lambda$; also the behavior of those weights as $\lambda$ approaches zero (or $\sigma$ grows unbounbed) is very specific: they become constant for linear regression and they grow up to infinity for logistic regression. In Section 4, we show (a) why these behaviors occur and (b) what the consequences are regarding the performance of the multiple kernel learning problem. In the remaining of this section, we review some important algorithmic issues[3].

**Step size selection**    A major issue in path following methods is the choice of the step $h$: if $h$ is too big, the predictor will end up very far from the path and many Newton steps have to be performed, while if $h$ is too small, progress is too slow. We chose a simple adaptive scheme where at each predictor step we select the biggest $h$ so that the predictor step stays in the domain $|J(\alpha, \sigma)| \leqslant \varepsilon$. The precision parameter $\varepsilon$ is itself adapted at each iteration: if the number of corrector steps at the previous iteration is greater than 8 then $\varepsilon$ is decreased whereas if this number is less than 4, it is increased.

**Running time complexity**    Between each kink, the path is smooth, thus there is a bounded number of steps [8, 9]. Each of those steps has complexity $O(n^3 + mn^2)$. We have observed empirically that the overall number of those steps is $O(m)$, thus the total empirical complexity is $O(mn^3 + m^2n^2)$. The complexity of solving the optimization problem in Eq. (5) using an interior point method for only one value of the regularization parameter is $O(mn^3)$ [2], thus if $m \leqslant n$, the empirical complexity of our algorithm, which yields the entire regularization path, is a constant times the complexity of obtaining only one point in the path using an interior point method. This makes intuitive sense, as both methods follow a path, by varying $\mu$ in the case of the interior point method, and by varying $\lambda$ in our case. The difference is that every point along our path is meaningful, not just the destination.

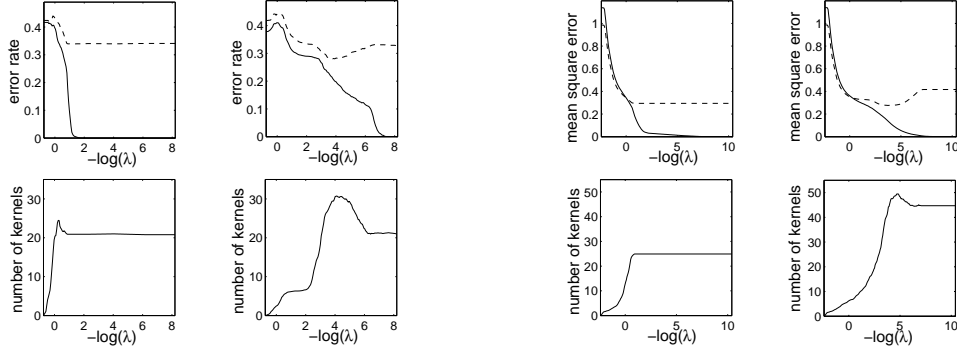

Figure 3: Varying the weights $(d_j)$: (left) classification on the Liver dataset, (right) regression on the Boston dataset ; for each dataset, two different values of $\gamma$, (left) $\gamma = 0$ and (right) $\gamma = 1$ . (Top) training set accuracy in bold, testing set accuracy in dashed, (bottom) number of kernels in the conic combination.

**Efficient implementation**    Because of our numerical regularization, none of the $\eta_j$'s are equal to zero (in fact each $\eta_j$ is lower bounded by $\mu/d_j^2$). We thus would have to use all kernels when computing the various derivatives. We circumvent this by truncating those $\eta_j$ that are close to their lower bound to zero: we thus only use the kernels that are numerically present in the combination.

**Second-order predictor step**    The implicit function theorem also allows to compute derivative of the path of higher orders. By using a second-order approximation of the path, we can reduce significantly the number of predictor-corrector steps required for the path.

## 4    Simulations

We have performed simulations on the Boston dataset (regression, 13 variables, 506 data points) and Liver dataset (classification, 6 variables, 345 data points) from the UCI repository, with the following kernels: linear kernel on all variables, linear kernels on single variables, polynomial kernels (with 4 different orders), Gaussian kernels on all variables (with 7 different kernel widths), Gaussian kernels on subsets of variables (also with 7 different kernel widths), and the identity matrix. This makes 110 kernels for the Boston dataset and 54 for the Liver dataset. All kernel matrices were normalized to unit trace.

Intuitively, the regularization weight $d_j$ for kernel $K_j$ should be an increasing function of the rank of $K_j$, i.e., we should penalize more feature spaces of higher dimensions. In order to explore the effect of $d_j$ on performance, we set $d_j$ as follows: we compute the number $p_j$ of eigenvalues of $K_j$ that are greater than $\frac{1}{2n}$ (remember that because of the unit trace constraint, these $n$ eigenvalues sum to 1), and we take $d_j = p_j^\gamma$. If $\gamma = 0$, then all $d_j$'s are equal to one, and when $\gamma$ increases, kernel matrices of high rank such as the identity matrix have relatively higher weights, noting that a higher weight implies a heavier regularization.

In Figure 3, for the Boston and liver datasets, we plot the number of kernels in the conic combination as well as the training and testing errors, for $\gamma = 0$ and $\gamma = 1$. We can make the following simple observations:

**Number of kernels**    The number of kernels present in the sparse conic combination is a non monotonic function of the regularization parameter. When the blocks are one-dimensional, a situation equivalent to variable selection with a 1-norm penalty, this number is usually a nearly monotonic function of the regularization parameter [4].

**Local minima**    Validation set performance may exhibit local minima, and thus algorithms

based on hill-climbing might exhibit poor performance by being trapped in a local minimum, whereas our approach where we compute the entire path would avoid that.

**Behavior for small** $\lambda$    For all values of $\gamma$, as $\lambda$ goes to zero, the number of kernels remains the same, the training error goes to zero, while the testing error remains constant. What changes when $\gamma$ changes is the value of $\lambda$ at which this behavior appears; in particular, for small values of $\gamma$, it happens before the testing error goes back up, leading to an unusual validation performance curve (an usual cross-validation curve would diverge to large values when the regularization parameter goes to zero). It is thus crucial to use weights $d_j$ that grow with the "size" of the kernel, and not simply constant.

This behavior can be confirmed by a detailed analysis of the optimality conditions, which show that if one of the kernel has a flat spectrum (such as the identity matrix), then, as $\lambda$ goes to zero, $\alpha$ tends to a limit, $\eta$ tends to a limit for linear regression and goes to infinity as $\log(1/\lambda)$ for logistic regression. Also, once in that limiting regime, the training error goes to zero quickly, while the testing error remains constant.

## 5   Conclusion

We have presented an algorithm to compute entire regularization paths for the problem of multiple kernel learning. Empirical results using this algorithm have provided us with insight into the effect of regularization for such problems. In particular we showed that the behavior of the block 1-norm regularization differs notably from traditional (non-block) 1-norm regularization.

As presented, the empirical results suggest that our algorithm scales quadratically in the number of kernels, but cubically in the number of data points. Indeed, the main computational burden (for both predictor and corrector steps) is the inversion of a Hessian. In order to make the computation of entire paths efficient for problems involving a large number of data points, we are currently investigating inverse Hessian updating, a technique which is commonly used in quasi-Newton methods [10].

### Acknowledgments

We wish to acknowledge support from NSF grant 0412995, a grant from Intel Corporation, and a graduate fellowship to Francis Bach from Microsoft Research.

## Footnotes

[1]We omit the intercept as it can be included by adding the constant variable equal to 1 to each feature vector $\Phi(x_i)$.

[2]In [1], the square of the block 1-norm was used. However, when the entire regularization path is sought, it is easy to show that the two problems are equivalent. The advantage of the current formulation is that when the blocks are of size one the problem reduces to classical 1-norm regularization [4].

[3]A Matlab implementation can be downloaded from www.cs.berkeley.edu/~fbach .

### References

[1] F. R. Bach, G. R. G. Lanckriet, and M. I. Jordan. Multiple kernel learning, conic duality, and the SMO algorithm. In *ICML*, 2004.

[2] G. R. G. Lanckriet, N. Cristianini, P. Bartlett, L. El Ghaoui, and M. I. Jordan. Learning the kernel matrix with semidefinite programming. *JMLR*, 5:27–72, 2004.

[3] C. S. Ong, A. J. Smola, and R. C. Williamson. Hyperkernels. In *NIPS 15*, 2003.

[4] B. Efron, T. Hastie, I. Johnstone, and R. Tibshirani. Least angle regression. *Ann. Stat.*, 32(2):407–499, 2004.

[5] T. Hastie, S. Rosset, R. Tibshirani, and J. Zhu. The entire regularization path for the support vector machine. In *NIPS 17*, 2005.

[6] A. Corduneanu and T. Jaakkola. Continuation methods for mixing heterogeneous sources. In *UAI*, 2002.

[7] T. Hastie, R. Tibshirani, and J. Friedman. *The Elements of Statistical Learning*. Springer-Verlag, 2001.

[8] E. L. Allgower and K. Georg. Continuation and path following. *Acta Numer.*, 2:1–64, 1993.

[9] S. Boyd and L. Vandenberghe. *Convex Optimization*. Cambridge Univ. Press, 2003.

[10] J. F. Bonnans, J. C. Gilbert, C. Lemaréchal, and C. A. Sagastizbal. *Numerical Optimization Theoretical and Practical Aspects*. Springer, 2003.
